# Performance Comparisons Between Backpropagation Networks and Classification Trees on Three Real-World Applications

**Les Atlas**
Dept. of EE, FT-10
University of Washington
Seattle, Washington 98195

**Ronald Cole**
Dept. of CS&E
Oregon Graduate Institute
Beaverton, Oregon 97006

**Jerome Connor, Mohamed El-Sharkawi, and Robert J. Marks II**
University of Washington

**Yeshwant Muthusamy**
Oregon Graduate Institute

**Etienne Barnard**
Carnegie-Mellon University

## ABSTRACT

Multi-layer perceptrons and trained classification trees are two very different techniques which have recently become popular. Given enough data and time, both methods are capable of performing arbitrary non-linear classification. We first consider the important differences between multi-layer perceptrons and classification trees and conclude that there is not enough theoretical basis for the clearcut superiority of one technique over the other. For this reason, we performed a number of empirical tests on three real-world problems in power system load forecasting, power system security prediction, and speaker-independent vowel identification. In all cases, even for piecewise-linear trees, the multi-layer perceptron performed as well as or better than the trained classification trees.

# 1   INTRODUCTION

In this paper we compare regression and classification systems. A regression system can generate an output $Y$ for an input $X$, where both $X$ and $Y$ are continuous and, perhaps, multi-dimensional. A classification system can generate an output class, C, for an input $X$, where $X$ is continuous and multi-dimensional and C is a member of a finite alphabet.

The statistical technique of Classification And Regression Trees (CART) was developed during the years 1973 (Meisel and Michalpoulos) through 1984 (Breiman *et al*). As we show in the next section, CART, like the multi-layer perceptron (MLP), can be trained to solve the exclusive-OR problem. Furthermore, the solution it provides is extremely easy to interpret. Moreover, both CART and MLPs are able to provide arbitrary piecewise linear decision boundaries. Although there have been no links made between CART and biological neural networks, the possible applications and paradigms used for MLP and CART are very similar.

The authors of this paper represent diverse interests in problems which have the commonality of being both important and potentially well-suited for trainable classifiers. The **load forecasting** problem, which is partially a regression problem, uses past load trends to predict the critical needs of future power generation. The **power security** problem uses the classifier as an interpolator of previously known states of the system. The **vowel recognition** problem is representative of the difficulties in automatic speech recognition due to variability across speakers and phonetic context.

In each problem area, large amounts of real data were used for training and disjoint data sets were used for testing. We were careful to ensure that the experimental conditions were identical for the MLP and CART. We concentrated only on performance as measured in error on the test set and did not do any formal studies of training or testing time. (CART was, in general, quite a bit faster.)

In all cases, even with various sizes of training sets, the multi-layer perceptron performed as well as or better than the trained classification trees. We also believe that integration of many of CART's well-designed attributes into MLP architectures could only improve the already promising performance of MLP's.

# 2   BACKGROUND

## 2.1   Multi-Layer Perceptrons

The name "artificial neural networks" has in some communities become almost synonymous with MLP's trained by back-propagation. Our power studies made use of this standard algorithm (Rumelhart *et al*, 1986) and our vowel studies made use of a conjugate gradient version (Barnard and Casasent, 1989) of back-propagation. In all cases the training data consisted of ordered pairs $\{(X,Y)\}$ for regression, or $\{(X,C)\}$ for classification. The input to the network is $X$ and the output is, after training, hopefully very close to $Y$ or C.

When MLP's are used for regression, the output, $Y$, can take on real values between 0 and 1. This normalized scale was used as the prediction value in the power forecasting problem. For MLP classifiers the output is formed by taking the (0,1) range of the output neurons and either thresholding or finding a peak. For example, in the vowel

study we chose the maximum of the 12 output neurons to indicate the vowel class.

## 2.2    Classification and Regression Trees (CART)

CART has already proven to be useful in diverse applications such as radar signal classification, medical diagnosis, and mass spectra classification (Breiman *et al*, 1984). Given a set of training examples $\{(X,C)\}$, a binary tree is constructed by sequentially partitioning the $p$-dimensional input space, which may consist of quantitative and/or qualitative data, into $p$-dimensional polygons. The trained classification tree divides the domain of the data into non-overlapping regions, each of which is assigned a class label C. For regression, the estimated function is piecewise constant over these regions.

The first split of the data space is made to obtain the best global separation of the classes. The next step in CART is to consider the partitioned training examples as two completely unrelated sets—those examples on the left of the selected hyper-plane, and those on the right. CART then proceeds as in the first step, treating each subset of the training examples independently. A question which had long plagued the use of such sequential schemes was: when should the splitting stop? CART implements a novel, and very clever approach; splits continue until every training example is separated from every other, then a pruning criterion is used to sequentially remove less important splits.

## 2.3    Relative Expectations of MLP and CART

The non-linearly separable exclusive-OR problem is an example of a problem which both MLP and CART can solve with zero error. The left side of Figure 1 shows a trained MLP solution to this problem and the right side shows the very simple trained CART solution. For the MLP the values along the arrows represent trained multiplicative weights and the values in the circles represent trained scalar offset values. For the CART figure, y and n represent yes or no answers to the trained thresholds and the values in the circles represent the output $Y$. It is interesting that CART did not train correctly for equal numbers of the four different input cases and that one extra example of one of the input cases was sufficient to break the symmetry and allow CART to train correctly. (Note the similarity to the well-known requirement of random and different initial weights for training the MLP).

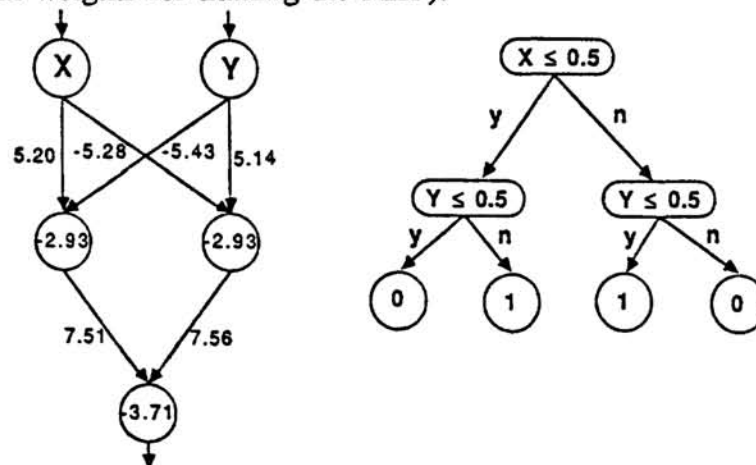

Figure 1:  The MLP and CART solutions to the exclusive-OR problem.

CART trains on the exclusive-OR very easily since a piecewise-linear partition in the input space is a perfect solution. In general, the MLP will construct classification regions with smooth boundaries, whereas CART will construct regions with "sharp" corners (each region being, as described previously, an intersection of half planes). We would thus expect MLP to have an advantage when classification boundaries tend to be smooth and CART to have an advantage when they are sharper.

Other important differences between MLP and CART include:

For an MLP the number of hidden units can be selected to avoid overfitting or underfitting the data. CART fits the complexity by using an automatic pruning technique to adjust the size of the tree. The selection of the number of hidden units or the tree size was implemented in our experiments by using data from a second training set (independent of the first).

An MLP becomes a classifier through an ad hoc application of thresholds or peak-picking to the output value(s). Great care has gone into the CART splitting rules while the usual MLP approach is rather arbitrary.

A trained MLP represents an approximate solution to an optimization problem. The solution may depend on initial choice of weights and on the optimization technique used. For complex MLP's many of the units are independently and simultaneously adjusting their weights to best minimize output error.

MLP is a distributed topology where a single point in the input space can have an effect across all units or analogously, one weight, acting alone, will have minimal affect on the outputs. CART is very different in that each split value can be mapped onto one segment in the input space. The behavior of CART makes it much more useful for data interpretation. A trained tree may be useful for understanding the structure of the data. The usefulness of MLP's for data interpretation is much less clear.

The above points, when taken in combination, do not make a clear case for either MLP or CART to be superior for the best performance as a trained classifier. We thus believe that the empirical studies of the next sections, with their consistent performance trends, will indicate which of the comparative aspects are the most significant.

## 3  LOAD FORECASTING

### 3.1  The Problem

The ability to predict electric power system loads from an hour to several days in the future can help a utility operator to efficiently schedule and utilize power generation. This ability to forecast loads can also provide information which can be used to strategically trade energy with other generating systems. In order for these forecasts to be useful to an operator, they must be accurate and computationally efficient.

### 3.2  Methods

Hourly temperature and load data for the Seattle/Tacoma area were provided for us by the Puget Sound Power and Light Company. Since weekday forecasting is a more critical problem for the power industry than weekends, we selected the hourly data for

all Tuesdays through Fridays in the interval of November 1, 1988 through January 31, 1989. These data consisted of 1368 hourly measurements that consisted of the 57 days of data collected.

These data were presented to both the MLP and the CART classifier as a 6-dimensional input with a single, real-valued output. The MLP required that all values be normalized to the range (0,1). These same normalized values were used with the CART technique. Our training and testing process consisted of training the classifiers on 53 days of the data and testing on the 4 days left over at the end of January 1989. Our training set consisted of 1272 hourly measurements and our test set contained 96 hourly readings.

The MLP we used in these experiments had 6 inputs (plus the trained constant bias term) 10 units in one hidden layer and one output. This topology was chosen by making use of data outside the training and test sets.

### 3.3    Results

We used an $l_1$ norm for the calculation of error rates and found that both techniques worked quite well. The average error rate for the MLP was 1.39% and CART gave 2.86% error. While this difference (given the number of testing points) is not statistically significant, it is worth noting that the trained MLP offers performance which is at least as good as the current techniques used by the Puget Sound Power and Light Company and is currently being verified for application to future load prediction.

## 4   POWER SYSTEM SECURITY

The assessment of security in a power system is an ongoing problem for the efficient and reliable generation of electric power. Static security addresses whether, after a disturbance, such as a line break or other rapid load change, the system will reach a steady state operating condition that does not violate any operating constraint and cause a "brown-out" or "black-out."

The most efficient generation of power is achieved when the power system is operating near its insecurity boundary. In fact, the ideal case for efficiency would be full knowledge of the absolute boundaries of the secure regions. Due to the complexity of the power systems, this full knowledge is impossible. Load flow algorithms, which are based on iterative solutions of nonlinearly constrained equations, are conventionally used to slowly and accurately determine points of security or insecurity. In real systems the trajectories through the regions are not predictable in fine detail. Also these changes can happen too fast to compute new results from the accurate load flow equations.

We thus propose to use the sparsely known solutions of the load flow equations as a training set. The test set consists of points of unknown security. The error of the test set can then be computed by comparing the result of the trained classifier to load flow equation solutions.

Our technique for converting this problem to a problem for a trainable classifier involves defining a training set {(X,C)} where X is composed of real power, reactive power, and apparent power at another bus. This 3-dimensional input vector is paired with the corresponding security status (C=1 for secure and C=0 for insecure). Since

the system was small, we were able to generate a large number of data points for training and testing. In fact, well over 20,000 total data points were available for the (disjoint) training and test sets.

## 4.1    Results

We observed that for any choice of training data set size, the error rate for the MLP was always lower than the rate for the CART classifier. At 10,000 points of training data, the MLP had an error rate of 0.78% and CART has an error rate of 1.46%. While both of these results are impressive, the difference was statistically significant ($p > .99$).

In order to gain insight into the reasons for differences in importance, we looked at classifier decisions for 2-dimensional slices of the input space. While the CART boundary sometimes was a better match, certain pathological difficulties made CART more error-prone than the MLP. Our other studies also showed that there were worse interpolation characteristics for CART, especially for sparse data. Apparently, starting with nonlinear combinations of inputs, which is what the MLP does, is better for the accurate fit than the stair-steps of CART.

## 5    SPEAKER-INDEPENDENT VOWEL CLASSIFICATION

Speaker-independent classification of vowels excised from continuous speech is a most difficult task because of the many sources of variability that influence the physical realization of a given vowel. These sources of variability include the length of the speaker's vocal tract, phonetic context in which the vowel occurs, speech rate and syllable stress.

To make the task even more difficult the classifiers were presented only with information from a single spectral slice. The spectral slice, represented by 64 DFT coefficients (0-4 kHz), was taken from the center of the vowel, where the effects of coarticulation with surrounding phonemes are least apparent.

The training and test sets for the experiments consisted of featural descriptions, $X$, paired with an associated class, C, for each vowel sample. The 12 monophthongal vowels of English were used for the classes, as heard in the following words: beat, bit, bet, bat, roses, the, but, boot, book, bought, cot, bird. The vowels were excised from the wide variety of phonetic contexts in utterances of the TIMIT database, a standard acoustic phonetic corpus of continuous speech, displaying a wide range of American dialectical variation (Fisher *et al*, 1986) (Lamel *et al*, 1986). The training set consisted of 4104 vowels from 320 speakers. The test set consisted of 1644 vowels (137 occurrences of each vowel) from a different set of 100 speakers.

The MLP consisted of 64 inputs (the DFT coefficients, each normalized between zero and one), a single hidden layer of 40 units, and 12 output units; one for each vowel category. The networks were trained using backpropagation with conjugate gradient optimization (Barnard and Casasent, 1989). The procedure for training and testing a network proceeded as follows: The network was trained on 100 iterations through the 4104 training vectors. The trained network was then evaluated on the training set and a different set of 1644 test vectors (the test set). The network was then trained for an additional 100 iterations and again evaluated on the training and test sets. This process was continued until the network had converged; convergence was observed as a

consistent decrease or leveling off of the classification percentage on the test data over successive sets of 100 iterations.

The CART system was trained using two separate computer routines. One was the CART program from California Statistical Software; the other was a routine we designed ourselves. We produced our own routine to ensure a careful and independent test of the CART concepts described in (Breiman *et al*, 1984).

## 5.1   Results

In order to better understand the results, we performed listening experiments on a subset of the vowels used in these experiments. The vowels were excised from their sentence context and presented in isolation. Five listeners first received training in the task by classifying 900 vowel tokens and receiving feedback about the correct answer on each trial. During testing, each listener classified 600 vowels from the test set (50 from each category) without feedback. The average classification performance on the test set was 51%, compared to chance performance of 8.3%. Details of this experiment are presented in (Muthusamy *et al*, 1990). When using the scaled spectral coefficients to train both techniques, the MLP correctly classified 47.4% of the test set while CART employing uni-variate splits performed at only 38.2%.

One reason for the poor performance of CART with uni-variate splits may be that each coefficient (corresponding to energy in a narrow frequency band) contains little information when considered independently of the other coefficients. For example, reduced energy in the 1 kHz band may be difficult to detect if the energy in the 1.06 kHz band was increased by an appropriate amount. The CART classifier described above operates by making a series of inquiries about one frequency band at a time, an intuitively inappropriate approach.

We achieved our best CART results, 46.4%, on the test set by making use of arbitrary hyper-planes (linear combinations) instead of univariate splits. This search-based approach gave results which were within 1% of the MLP results.

## 6   CONCLUSIONS

In all cases the performance of the MLP was, in terms of percent error, better than CART. However, the difference in performance between the two classifiers was only significant (at the $p > .99$ level) for the power security problem.

There are several possible reasons for the sometimes superior performance of the MLP technique, all of which we are currently investigating. One advantage may stem from the ability of MLP to easily find correlations between large numbers of variables. Although it is possible for CART to form arbitrary nonlinear decision boundaries, the efficiency of the recursive splitting process may be inferior to MLP's nonlinear fit. Another relative disadvantage of CART may be due to the successive nature of node growth. For example, if the first split that is made for a problem turns out, given the successive splits, to be suboptimal, it becomes very inefficient to change the first split to be more suitable.

We feel that the careful statistics used in CART could also be advantageously applied to MLP. The superior performance of MLP is not yet indicative of best performance and it may turn out that careful application of statistics may allow further advance-

ments in the MLP technique. It also may be possible that there would be input representations that would cause better performance for CART than for MLP.

There have been new developments in trained statistical classifiers since the development of CART. More recent techniques, such as projection pursuit (Friedman and Stuetzle, 1984), may prove as good as or superior to MLP. This continued interplay between MLP techniques and advanced statistics is a key part of our ongoing research.

### Acknowledgements

The authors wish to thank Professor R.D. Martin and Dr. Alan Lippman of the University of Washington Department of Statistics and Professors Aggoune, Damborg, and Hwang of the University of Washington Department of Electrical Engineering for their helpful discussions. David Cohn and Carlos Rivera assisted with many of the experiments.

We also would like to thank Milan Casey Brace of Puget Power and Light for providing the load forecasting data.

This work was supported by a National Science Foundation Presidential Young Investigator Award for L. Atlas and also by separate grants from the National Science Foundation and Washington Technology Center.

### References

P. E. Barnard and D. Casasent, "Image Processing for Image Understanding with Neural Nets," *Proc. Int. Joint Conf. on Neural Nets*, Washington, DC, June 18-22, 1989.

L. Breiman, J.H. Friedman, R.A. Olshen, and C.J. Stone, **Classification and Regression Trees,** Wadsworth International, Belmont, CA, 1984.

W. Fisher, G. Doddington, and K. Goudie-Marshall, "The DARPA Speech Recognition Research Database: Specification and Status," *Proc. of the DARPA Speech Recognition Workshop,* pp. 93-100, February 1986.

J.H. Friedman and W. Stuetzle, "Projection Pursuit Regression," *J. Amer. Stat. Assoc.* **79**, pp. 599-608, 1984.

L. Lamel, R. Kassel, and S. Seneff, "Speech Database Development: Design and Analysis of the Acoustic-Phonetic Corpus," *Proc. of the DARPA Speech Recognition Workshop*, pp. 100-110, February 1986.

W.S. Meisel and D.A. Michalpoulos, "A Partitioning Algorithm with Application in Pattern Classification and the Optimization of Decision Trees," *IEEE Trans. Computers* C-22, pp. 93-103, 1973.

Y. Muthusamy, R. Cole, and M. Slaney, "Vowel Information in a Single Spectral Slice: Cochleagrams Versus Spectrograms," *Proc. ICASSP '90*, April 3-6, 1990. (to appear)

D.E. Rumelhart, G.E. Hinton, and R.J. Williams, "Learning Internal Representations by Error Propagation," Ch. 2 in **Parallel Distributed Processing**, D.E. Rumelhart, J.L. McClelland, and the PDP Research Group, MIT Press, Cambridge, MA, 1986.
